# Dependent Dirichlet Process Spike Sorting

**Jan Gasthaus, Frank Wood, Dilan Görür, Yee Whye Teh**
Gatsby Computational Neuroscience Unit
University College London
London, WC1N 3AR, UK
{j.gasthaus, fwood, dilan, ywteh}@gatsby.ucl.ac.uk

## Abstract

In this paper we propose a new incremental spike sorting model that automatically eliminates refractory period violations, accounts for action potential waveform drift, and can handle "appearance" and "disappearance" of neurons. Our approach is to augment a known time-varying Dirichlet process that ties together a sequence of infinite Gaussian mixture models, one per action potential waveform observation, with an interspike-interval-dependent likelihood that prohibits refractory period violations. We demonstrate this model by showing results from sorting two publicly available neural data recordings for which a partial ground truth labeling is known.

## 1 Introduction

Spike sorting (see [1] and [2] for review and methodological background) is the name given to the problem of grouping action potentials by source neuron. Generally speaking, spike sorting involves a sequence of steps; 1) recording the activity of an unknown number of neurons using some kind of extra-cellular recording device, 2) detecting the times at which action potentials are likely to have occurred, 3) slicing action potential waveforms from the surrounding raw voltage trace where action potentials were posited to have occurred, 4) (often) performing some kind of dimensionality reduction/feature extraction on the set of collected action potential waveform snippets, 5) running a clustering algorithm to produce grouping of action potentials attributed to a single neuron, and finally 6) running some kind of post hoc algorithm that detects refractory period violations and thins or adjusts the clustering results accordingly.

Neuroscientists are interested in arriving at *the* optimal solution to this problem. Towards this end they have traditionally utilized maximum likelihood clustering methods such as expectation maximization for finite Gaussian mixture models with cross-validation model selection. This of course allows them to arrive at *an* optimal solution, but it is difficult to say whether or not it is *the* optimal solution, and it affords them no way of establishing the level of confidence they should have in their result. Recently several groups have suggested a quite different approach to this problem which eschews the quest for a single optimal solution in favor of a Bayesian treatment of the problem [3, 4, 5, 6]. In each of these, instead of pursuing the optimal sorting, multiple sortings of the spikes are produced (in fact what each model produces is a posterior distribution over spike trains). Neural data analyses may then be averaged over the resulting spike train distribution to account for uncertainties that may have arisen at various points in the spike sorting process and would not have been explicitly accounted for otherwise.

Our work builds on this new Bayesian approach to spike sorting; going beyond them in the way steps five and six are accomplished. Specifically we apply the generalized Polya urn dependent Dirichlet process mixture model (GPUDPM) [7, 8] to the problem of spike sorting and show how it allows us to model waveform drift and account for neuron appearance and disappearance. By introducing a time dependent likelihood into the model we are also able to eliminate refractory period violations.

The need for a spike sorting approach with these features arises from several domains. Waveform non-stationarities either due to changes in the recording environment (e.g. movement of the electrode) or due to changes in the firing activity of the neuron itself (e.g. burstiness) cause almost all current spike sorting approaches to fail. This is because most pool waveforms over time, discarding the time at which the action potentials were observed. A notable exception to this is the spike sorting approach of [9], in which waveforms were pooled and clustered in short fixed time intervals. Multiple Gaussian mixture models are then fit to the waveforms in each interval and then are pruned and smoothed until a single coherent sequence of mixture models is left that describes the entire time course of the data. This is accomplished by using a forward-backward-like algorithm and the Jenson-Shannon divergence between models in consecutive intervals. Although very good results can be produced by such a model, using it requires choosing values for a large number of parameters, and, as it is a smoothing algorithm, it requires the entire data set to have been observed already.

A recent study by [10] puts forward a compelling case for online spike sorting algorithms that can handle waveform non-stationarity, as well as sudden jumps in waveform shape (e.g. abrupt electrode movements due to high acceleration events), and appearance and disappearance of neurons from the recording over time. This paper introduces a chronical recording paradigm in which a chronically implanted recording device is mated with appropriate storage such that very long term recordings can be made. Unfortunately as the animal being recorded from is allowed its full range of natural movements, accelerations may cause the signal characteristics of the recording to vary dramatically over short time intervals. As such data theoretically can be recorded forever without stopping, forward-backward spike sorting algorithms such as that in [9] are ruled out. As far as we know our proposed model is the only sequential spike sorting model that meets all of the requirements of this new and challenging spike sorting problem,

In the next sections we review the GPUDPM on which our spike sorting model is based, introduce the specifics of our spike sorting model, then demonstrate its performance on real data for which a partial ground truth labeling is known.

## 2 Review

Our model is based on the generalized Polya urn Dirichlet process mixture model (GPUDPM) described in [7, 8]. The GPUDPM is a time dependent Dirichlet process (DDP) mixture model formulated in the Chinese restaurant process (CRP) sampling representation of a Dirichlet process mixture model (DPM). We will first very briefly review DPMs in general and then turn to the specifics of the GPUDPM.

DPMs are a widely used tool for nonparametric density estimation and unsupervised learning in models where the true number of latent classes is unknown. In a DPM, the mixing distribution $\mathbb{G}$ is distributed according to a DP with base distribution $\mathbb{G}_0$, i.e.

$$
\begin{array}{rcl}
\mathbb{G}|\alpha, \mathbb{G}_0 & \sim & \mathrm{DP}(\alpha, \mathbb{G}_0) \\
\theta_i|\mathbb{G} & \sim & \mathbb{G} \\
\mathbf{x}_i|\theta_i & \sim & F(\theta_i)
\end{array}
\tag{1}
$$

Placing a DP prior over $\mathbb{G}$ induces a clustering tendency amongst the $\theta_i$. If $\theta_i$ takes on $K$ distinct values $\phi_1, \ldots, \phi_K$, we can write out an equivalent model using indicator variables $c_i \in \{1, \ldots, K\}$ that assigns data points to clusters. In this representation we track the distinct $\phi_k$ drawn from $\mathbb{G}_0$ for each cluster, and use the Chinese restaurant process to sample the conditional distributions of the indicator variables $c_i$

$$
\begin{array}{rcl}
P(c_i = k|c_1, \ldots, c_{i-1}) & = & \frac{m_k}{i-1+\alpha} \quad \text{for } k \in \{c_j : j < i\} \\
P(c_i \neq c_j \text{ for all } j < i|c_1, \ldots, c_{i-1}) & = & \frac{\alpha}{i-1+\alpha}
\end{array}
\tag{2}
$$

where $m_k = \#\{c_j : c_j = k \wedge j < i\}$.

The GPUDPM consists of $T$ individual DPMs, one per discrete time step $t = 1, \ldots, T$, all tied together through a particular way of sharing the component parameters $\phi_k^t$ and table occupancy counts $m_k^t$ between adjacent time steps (here $t$ indexes the parameters and cluster sizes of the $T$ DPMs).

Dependence among the $m_k^t$ is introduced by perturbing the number of customers sitting at each table when moving forward through time. Denote by $\mathbf{m}^t = (m_1^t, \ldots, m_{K^t}^t)$ the vector containing the

number of customers sitting at each table at time $t$ before a "deletion" step, where $K^t$ is the number of non-empty tables at time $t$. Similarly denote by $\mathbf{m}^{t+1}$ the same quantity after this deletion step. Then the perturbation of the class counts from one step to the next is governed by the process

$$\mathbf{m}^{t+1}|\mathbf{m}^t, \rho \sim \begin{cases} \mathbf{m}^t - \boldsymbol{\xi}^t & \text{with probability } \gamma \\ \mathbf{m}^t - \boldsymbol{\zeta}^t & \text{with probability } 1-\gamma \end{cases} \tag{3}$$

where $\xi_k^t \sim \text{Binomial}(m_k^t, 1 - \rho)$ and $\zeta_j^t = m_j^t$ for $j \neq \ell$ and $\zeta_j^t = 0$ for $j = \ell$ where $\ell \sim \text{Discrete}(\mathbf{m}^t / \sum_{k=1}^{K^t} m_k^t)$. Before seating the customers arriving at time step $t+1$, the number of customers sitting at each table is initialized to $\mathbf{m}^{t+1}$. This perturbation process can either remove some number of customers from a table or effectively delete a table altogether. This deletion procedure accounts for the ability of the GPUDPM to model births and deaths of clusters.

The GPUDPM is also capable of modeling drifting cluster parameters. This drift is modeled by tying together the component parameters $\phi_k^t$ through a transition kernel $\text{P}(\phi_k^t|\phi_k^{t-1})$ from which the class parameter at time $t$ is sampled given the class parameter at time $t-1$. For various technical reasons one must ensure that the mixture component parameters $\phi_k^t$ are all drawn independently from $\mathbb{G}_0$, i.e. $\{\phi_k^t\}_{t=1}^T \sim \mathbb{G}_0$. This can be achieved by ensuring that $\mathbb{G}_0$ is the invariant distribution of the transition kernel $\text{P}(\phi_k^t|\phi_k^{t-1})$ [8].

## 3 Model

In order to apply the GPUDPM model to spike sorting problems one first has to make a number of modeling assumptions. First is choosing a form for the likelihood function describing the distribution of action potential waveform shapes generated by a single neuron $P(\mathbf{x}^t|c^t = k, \theta_k^t)$ (the distibution of which was denoted $F(\theta_k^t)$ above), the prior over the parameters of that model (the base distribution $\mathbb{G}_0$ above), and the transition kernel $\text{P}(\phi_k^t|\phi_k^{t-1})$ that governs how the waveshape of the action potentials emitted by a neuron can change over time. In the following we describe modeling choices we made for the spike sorting task, as well as how the continuous spike occurrence times can be incorporated into the model to allow for correct treatment of neuron behaviour during the absolute refractory period.

Let $\{\mathbf{x}^t\}_{t=1}^T$ be the the set of action potential waveforms extracted from an extracellular recording (referred to as "spikes" in the following), and let $\tau^1, \ldots, \tau^T$ be the time stamps (in ms) associated with these spikes in ascending order (i.e. $\tau^t \geq \tau^{t'}$ if $t > t'$). The model thus incorporates two different concepts of time: the discrete sequence of time steps $t = 1, \ldots, T$ corresponding to the time steps in the GPUDPM model and the actual spike times $\tau^t$ at which the spike $\mathbf{x}^t$ occurs in the recording. We assume that only one spike occurs per time step $t$, i.e. we set $N = 1$ in the model above and identify $\mathbf{c}^t = (c_1^t) = c^t$.

It is well known that the distribution of action potential waveforms originating from a single neuron in a PCA feature space is well approximated by a Normal distribution [1]. We choose to model each dimension $x_d$ ($d \in \{1, \ldots, D\}$) of the data independently with a univariate Normal distribution and use a product of independent Normal-Gamma priors as the base distribution $\mathbb{G}_0$ of the DP.

$$P(\mathbf{x}|\phi) \overset{\text{def}}{=} \mathcal{N}(\mathbf{x}|\phi) = \prod_{d=1}^D \mathcal{N}\left(x_d|\mu_d, \lambda_d^{-1}\right) \tag{4}$$

$$\mathbb{G}_0(\boldsymbol{\mu}_0, n_0, a, b) \overset{\text{def}}{=} \prod_{d=1}^D \left[\mathcal{N}\left(\mu_d|\mu_{0,d}, (n_0\lambda_d)^{-1}\right) \text{Ga}\left(\lambda_d|a, b\right)\right] \tag{5}$$

where $\phi = (\lambda_1, \ldots, \lambda_D, \mu_1, \ldots, \mu_D)$, and $\boldsymbol{\mu}_0 = (\mu_{0,1}, \ldots, \mu_{0,D})$, $n_0$, $a$, and $b$ are parameters of the model. The independence assumption is made here mainly to increase computational efficiency. A model where $P(\mathbf{x}|\phi)$ is a multivariate Gaussian with full covariance matrix is also possible, but makes sampling from (7) computationally expensive. While correlations between the components can be observed in neural recordings, they can at least partially be attributed to temporal waveform variation.

To account for the fact that neurons have an absolute refractory period following each action potential during which no further action potential can occur, we extend the GPUDPM by conditioning the model

on the spike occurrence times $\tau_1, \ldots, \tau_T$ and modifying the conditional probability of assigning a spike to a cluster given the other cluster labels and the spike occurrence times $\tau_1, \ldots, \tau_t$ in the following way:

$$\mathrm{P}(c_t = k | \mathbf{m}^t, c_{1:t-1}, \tau^{1:t}, \alpha) \propto \begin{cases} 0 & \text{if } \tau^t - \hat{\tau}_k^t \leq r_{\mathrm{abs}} \\ m_k^t & \text{if } \tau^t - \hat{\tau}_k^t > r_{\mathrm{abs}} \text{ and } k \in \{1, \ldots, K_{t-1}\} \\ \alpha & \text{if } \tau^t - \hat{\tau}_k^t > r_{\mathrm{abs}} \text{ and } k = K_{t-1} + 1 \end{cases} \quad (6)$$

where $\hat{\tau}_k^t$ is the spike time of the last spike assigned to cluster $k$ before time step $t$, i.e. $\hat{\tau}_k^t = \tau^{t'}$, $t' = \max\{t'' | t'' < t \wedge c_{t''} = k\}$. Essentially, the conditional probability of assigning the spike at time $t$ to cluster $k$ is zero if the difference of the occurrence time of this spike and the occurrence time of the last spike associated with cluster $k$ is smaller than the refractory period $r_{\mathrm{abs}}$. If the time difference is larger than $r_{\mathrm{abs}}$ then the usual CRP conditional probabilities are used. In terms of the Chinese restaurant metaphor, this setup corresponds to a restaurant in which seating a customer at a table removes that table as an option for new customers for some period of time. Note that this extension introduces additional dependencies among the indicator variables $c_1, \ldots, c_T$.

The transition kernel $\mathrm{P}(\phi_k^t | \phi_k^{t-1})$ specifies how the action potential waveshape can vary over time. To meet the technical requirements of the GPUDPM and because its waveform drift modeling semantics are reasonable we use the update rule of the Metropolis algorithm [11] as the transition kernel $\mathrm{P}(\phi_k^t | \phi_k^{t-1})$, i.e. we set

$$\mathrm{P}(\phi_k^t | \phi_k^{t-1}) = S(\phi_k^{t-1}, \phi_k^t) A(\phi_k^{t-1}, \phi_k^t) + \left(1 - \int S(\phi', \phi_k^t) A(\phi', \phi_k^t) d\phi'\right) \delta_{\phi_k^{t-1}}(\phi_k^t) \quad (7)$$

where $S(\phi', \phi_k^t)$ is a (symmetric) proposal distribution and $A(\phi', \phi_k^t) = \min\left(1, \mathbb{G}_0(\phi_k^t)/\mathbb{G}_0(\phi_k^{t-1})\right)$. We choose an isotropic Gaussian centered at the old value as proposal distribution $S(\phi', \phi_k^t) = \mathcal{N}(\phi_k^{t-1}, \sigma \mathbf{I})$. This choice of $P(\phi_k^t | \phi_k^{t-1})$ ensures that $\mathbb{G}_0$ is the invariant distribution of the transition kernel, while at the same time allowing us to control the amount of correlation between time steps through $\sigma$. A transition kernel of this form allows the distribution of the action potential waveforms to vary slowly (if $\sigma$ is chosen small) from one time step to the next both in mean waveform shape as well as in variance. While small changes are preferred, larger changes are also possible if supported by the data.

Inference in this model is performed using the sequential Monte Carlo algorithm (particle filter) defined in [7, 8].

## 4 Experiments

### 4.1 Methodology

Experiments were performed on a subset of the publicly available[1] data set described in [12, 13], which consists of simultaneous intracellular and extracellular recordings of cells in the hippocampus of anesthetized rats. Recordings from an extracellular tetrode and an intracellular electrode were made simultaneously, such that the cell recorded on the intracellular electrode was also recorded extracellularly by a tetrode.

Action potentials detected on the intracellular (IC) channel are an almost certain indicator that the cell being recorded spiked. Action potentials detected on the extracellular (EC) channels may include the action potentials generated by the intracellularly recorded cell, but almost certainly include spiking activity from other cells as well. The intracellular recording therefore can be used to obtain a ground truth labeling for the spikes originating from one neuron that can be used to evaluate the performance of human sorters and automatic spike sorting algorithms that sort extracellular recordings [13]. However, by this method ground truth can only be determined for one of the neurons whose spikes are present in the extracellular recording, and this should be kept in mind when evaluating the performance of spike sorting algorithms on such a data set. Neither the correct number of distinct neurons recorded from by the extracellular electrode nor the correct labeling for any spikes not originating from the neuron recorded intracellularly can be determined by this methodology.

| Data set | | DPM | | | GPUDPM | | |
|---|---|---|---|---|---|---|---|
| | | FP | FN | RPV | FP | FN | RPV |
| 1 | MAP | 4.90% | 4.21% | 4 | 4.71% | 1.32% | 0 |
| | AVG | 5.11% | 5.17% | 4 | 4.77% | 1.68% | 0 |
| 2 | MAP | 0.94% | 9.40% | 1 | 0.85% | 18.63% | 0 |
| | AVG | 0.83% | 12.48% | 1 | 0.86% | 18.81% | 0 |

Table 1: Performance of both algorithms on the two data sets: % false positives (FP), % false negatives (FN), # of refratory period violations (RPV). Results are shown for the MAP solution (MAP) and averaged over the posterior distribution (AVG).

The subset of that data set that was used for the experiments consisted of two recordings from different animals (4 minutes each), recorded at 10 kHz. The data was bandpass filtered (300Hz – 3kHz), and spikes on the intracellular channel were detected as the local maxima of the first derivative of the signal larger than a manually chosen threshold. Spikes on the extracellular channels were determined as the local minima exceeding 4 standard deviations in magnitude. Spike waveforms of length 1 ms were extracted from around each spike (4 samples before and 5 samples after the peak). The positions of the minima within the spike waveforms were aligned by upsampling, shifting and then downsampling the waveforms. The extracellular spikes corresponding to action potentials from the identified neuron were determined as the spikes occurring within 0.1 ms of the IC spike.

For each spike the signals from the four tetrode channels were combined into a vector of length 40. Each dimensions was scaled by the maximal variance among all dimensions and PCA dimensionality reduction was performed on the scaled data sets (for each of the two recordings separately). The first three principal components were used as input to our spike sorting algorithm. The first recording (data set 1) consists of 3187 spikes, 831 originate from the identified neuron, while the second (data set 2) contains 3502 spikes, 553 of which were also detected on the IC channel. As shown in Figure 1, there is a clearly visible change in waveform shape of the identified neuron over time in data set 1, while in data set 2 the waveform shapes remain roughly constant. Presumably this change in waveform shape is due to the slow death of the cell as a result of the damage done to the cell by the intracellular recording procedure.

The parameters for the prior ($\boldsymbol{\mu}_0$, $n_0$, $a$, $b$) were chosen empirically and were fixed at $\boldsymbol{\mu}_0 = \mathbf{0}$, $n_0 = 0.1$, $a = 4$, $b = 1$ for all experiments. The parameters governing the deletion procedure were set to $\rho = 0.985$ and $\gamma = 1 - 10^{-5}$, reflecting the fact that we consider relative firing rates of the neurons to stay roughly constant over time and neuron death a relatively rare process respectively. The variance of the proposal distribution $\sigma$ was fixed at $0.01$, favoring small changes in the cluster parameters from one time step to the next. Experiments on both data sets were performed for $\alpha \in \{0.01, 0.005, 0.001\}$ and the model was found to be relatively sensitive to this parameter in our experiments. The sequential Monte Carlo simulations were run using 1000 particles, and multinomial resampling was performed at each step.

For comparison, the same data set was also sorted using the DPM-based spike sorting algorithm described in [6][2], which pools waveforms over time and thus does not make use of any information about the occurrence times of the spikes. The algorithm performs Gibbs sampling in a DPM with Gaussian likelihood and a conjugate Normal-Inverse-Wishart prior. A Gamma prior is placed on the DP concentration parameter $\alpha$. The parameters of the priors the prior were set to $\boldsymbol{\mu}_0 = 0$, $\kappa_0 = 0.1$, $\lambda_0 = 0.1 \times \mathbf{I}$, $a_0 = 1$ and $b_0 = 1$. The Gibbs sampler was run for 6000 iterations, where the first 1000 were discarded as burn-in.

### 4.2 Results

The performance of both algorithms is shown in Table 1. The data labelings corresponding to these results are illustrated in Figure 1. As expected, our algorithm outperforms the DPM-based algorithm on data set 1, which includes waveform drift which the DPM cannot account for. As data set 2 does not show waveform drift it can be adequately modeled without introducing time dependence. The DPM model which has the advantage of being significantly less complex than the GPUDPM is able

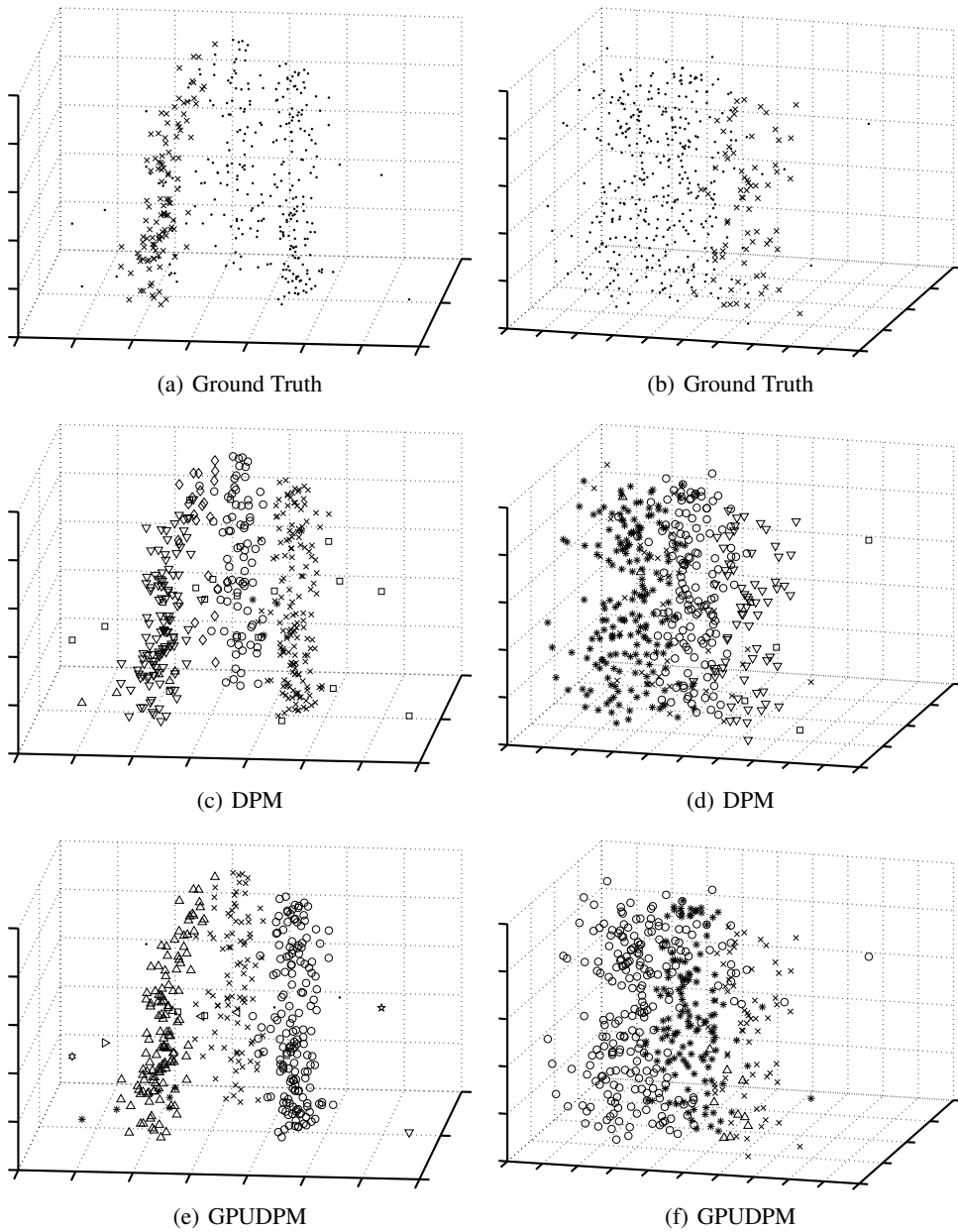

(a) Ground Truth　　　　　　　　　　(b) Ground Truth

(c) DPM　　　　　　　　　　　　　(d) DPM

(e) GPUDPM　　　　　　　　　　(f) GPUDPM

Figure 1: A comparison of DPM to GPUDPM spike sorting for two channels of tetrode data for which the ground truth labeling of one neuron is known. Each column shows subsampled results for one data set. In all plots the vertical axis is time and the horizontal axes are the first two principal components of the detected waveforms. The top row of graphs shows the ground truth labeling of both data sets where the action potentials known to have been generated by a single neuron are labeled with x's. Other points in the top row of graphs may also correspond to action potentials but as we do not know the ground truth labeling for them we label them all with dots. The middle row shows the maximum a posteriori labeling of both data sets produced by a DP mixture model spike sorting algorithm which does not utilize the time at which waveforms were captured, nor does it model waveform shape change. The bottom row shows the maximum a posteriori labeling of both data sets produced by our GPUDPM spike sorting algorithm which does model both the time at which the spikes occurred and the changing action potential waveshape. The left column shows that the GPUDPM performs better than the DPM when the waveshape of the underlying neurons changes over time. The right column shows that the GPUDPM performs no worse than the DPM when the waveshape of the underlying neurons stays constant.

to outperform our model on this data set. The inferior performance of the GPUDPM model on this data set can also partly be be explained by the inference procedure used: For the GPUDPM model inference is performed by a particle filter using a relatively small number of particles (1000), whereas a large number of Gibbs sampler iterations (5000) are used to estimate the posterior for the DPM. With a larger number of particles (or samples in the Gibbs sampler), one would expect both models to perform equally well, with possibly a slight advantage for the GPUDPM which can exploit the information contained in the refractory period violations. As dictated by the model, the GPUDPM algorithm does not assign two spikes that are within the refractory period of each other to the same cluster, whereas the DPM does not incorporate this restriction, and therefore can produce labelings containing refractory period violations. Though only a relatively small number of such mistakes are made by the DPM algorithm, these effects are likely to become larger in longer and/or noisier recordings, or when more neurons are present.

For some values of $\alpha$ the GPUDPM algorithm produced different results, showing either a large number of false positives or a large number of false negatives. In the former case the algorithm incorrectly places the waveforms from the IC channel and the waveform of another neuron in one cluster, in the latter case the algorithm starts assigning the IC waveforms to a different cluster after some point in time. This behavior is illustrated for data set 1 and $\alpha = 0.01$ in Figure 2, and can be explained by shortcomings of the inference scheme: While in theory the algorithm should be able to maintain multiple labeling hypotheses throughout the entire time span, the particle filter approach – especially when the number of particles is small and no specialized resampling scheme (e.g. [14]) is used – in practice often only represents the posterior accurately for the last few time steps.

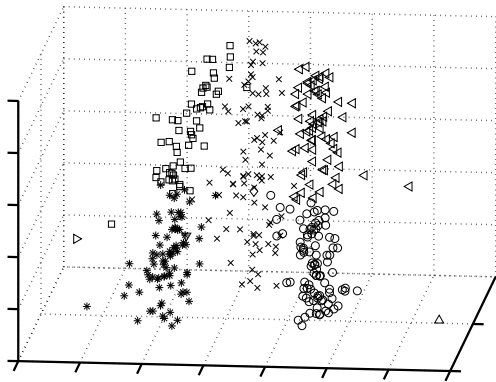

Figure 2: An alternative "interpretation" of the data from the left column of Fig. 1 given by the GPUDPM spike sorter. Here the labels assigned to both the the neuron with changing waveshape and one of the neurons with stationary waveshape change approximately half-way through the recording. Although it is difficult to see because the data set must be significantly downsampled for display purposes, there is a "noise event" at the point in time where the labels switch. A feature of the DDP is that it assigns posterior mass to both of these alternative interpretations of the data. While for this data set we know this labeling to be wrong because we know the ground truth, in other recordings such an "injection of noise" could, for instance, signal a shift in electrode position requiring similar rapid births and deaths of clusters.

## 5   Discussion

We have demonstrated that spike sorting using time-varying Dirichlet process mixtures in general, and more specifically our spike sorting specialization of the GPUDPM, produce promising results. With such a spike sorting approach we, within a single model, are able to account for action potential waveform drift, refractory period violations, and neuron appearance and disappearance from a recording. Previously no single model addressed all of these simultaneously, requiring solutions in the form of ad hoc combinations of strategies and algorithms that produces spike sorting results that were potentially difficult to characterize. Our model-based approach makes it easy to explicitly state modeling assumptions and produces results that are easy to characterize. Also, more complex or application specific models of the interspike interval distribution and/or the data likelihood can easily

be incorporated into the model. The performance of the model on real data suggests that a more complete characterization of its performance is warranted. Directions for further research include the development of a more efficient sequential inference scheme or a hybrid sequential/Gibbs sampler scheme that allows propagation of interspike interval information backwards in time. Parametric models for the interspike interval density for each neuron whose parameters are inferred from the data, which can improve spike sorting results [15], can also be incorporated into the model. Finally, priors may be placed on some of the parameters in order to make make the algorithm more robust and easily applicable to new data.

### Acknowledgments

This work was supported by the Gatsby Charitable Foundation and the PASCAL Network of Excellence.

## Footnotes

[1] http://crcns.org/data-sets/hc/hc-1/

[2]Code publicly available from `http://www.gatsby.ucl.ac.uk/~fwood/code.html`

### References

[1] M. S. Lewicki. A review of methods for spike sorting: the detection and classification of neural action potentials. *Network: Computation in Neural Systems*, 9(4):53–78, 1998.

[2] M. Sahani. *Latent variable models for neural data analysis*. PhD thesis, California Institute of Technology, Pasadena, California, 1999.

[3] D. P. Nguyen, L. M. Frank, and E. N. Brown. An application of reversible-jump Markov chain Monte Carlo to spike classification of multi-unit extracellular recordings. *Network*, 14(1):61–82, 2003.

[4] D. Görür, C. R. Rasmussen, A. S. Tolias, F. Sinz, and N.K. Logothetis. Modeling spikes with mixtures of factor analyzers. In *Proceeding of the DAGM Symposium*, pages 391–398. Springer, 2004.

[5] F. Wood, S. Goldwater, and M. J. Black. A non-parametric Bayesian approach to spike sorting. In *Proceedings of the 28th Annual International Conference of the IEEE Engineering in Medicine and Biology Society*, pages 1165–1168, 2006.

[6] F. Wood and M. J. Black. A nonparametric Bayesian alternative to spike sorting. *Journal of Neuroscience Methods*, 173:1–12, 2008.

[7] F. Caron. *Inférence Bayésienne pour la détermination et la sélection de modèles stochastiques*. PhD thesis, École Centrale de Lille and Université des Sciences et Technologiques de Lille, Lille, France, 2006.

[8] F. Caron, M. Davy, and A. Doucet. Generalized Polya urn for time-varying Dirichlet process mixtures. In *23rd Conference on Uncertainty in Artificial Intelligence (UAI'2007), Vancouver, Canada, July 2007*, 2007.

[9] A. Bar-Hillel, A. Spiro, and E. Stark. Spike sorting: Bayesian clustering of non-stationary data. *Journal of Neuroscience Methods*, 157(2):303–316, 2006.

[10] G. Santhanam, M. D. Linderman, V. Gilja, A. Afshar, S. I. Ryu, T. H. Meng, and K. V. Shenoy. HermesB: A continuous neural recording system for freely behaving primates. *IEEE Transactions on Biomedical Engineering*, 54(11):2037–2050, 2007.

[11] A. W. Metropolis, A. W. Rosenbluth, M. N. Rosenbluth, A. H. Teller, and E. Teller. Equations of state calculations by fast computing machines. *Journal of Chemical Physics*, 21:1087–1092, 1953.

[12] D. A. Henze, Z. Borhegyi, J. Csicsvari, A. Mamiya, K. D. Harris, and G. Buzsáki. Intracellular features predicted by extracellular recordings in the hippocampus in vivo. *Journal of Neurophysiology*, 84(1):390–400, 2000.

[13] K. D. Harris, D. A. Henze, J. Csicsvari, H. Hirase, and G. Buzsáki. Accuracy of tetrode spike separation as determined by simultaneous intracellular and extracellular measurements. *Journal of Neurophysiology*, 81(1):401–414, 2000.

[14] P. Fearnhead. Particle filters for mixture models with an unknown number of components. *Journal of Statistics and Computing*, 14:11–21, 2004.

[15] C. Pouzat, M. Delescluse, P. Viot, and J. Diebolt. Improved spike-sorting by modeling firing statistics and burst-dependent spike amplitude attenuation: A Markov Chain Monte Carlo approach. *Journal of Neurophysiology*, 91(6):2910–2928, 2004.

